# From Batch to Transductive Online Learning

**Sham Kakade**
Toyota Technological Institute
Chicago, IL 60637
sham@tti-c.org

**Adam Tauman Kalai**
Toyota Technological Institute
Chicago, IL 60637
kalai@tti-c.org

## Abstract

It is well-known that everything that is learnable in the difficult online setting, where an arbitrary sequences of examples must be labeled one at a time, is also learnable in the batch setting, where examples are drawn independently from a distribution. We show a result in the opposite direction. We give an efficient conversion algorithm from batch to online that is transductive: it uses future unlabeled data. This demonstrates the equivalence between what is properly and *efficiently* learnable in a batch model and a transductive online model.

## 1 Introduction

There are many striking similarities between results in the standard batch learning setting, where labeled examples are assumed to be drawn independently from some distribution, and the more difficult online setting, where labeled examples arrive in an arbitrary sequence. Moreover, there are simple procedures that convert any online learning algorithm to an equally good batch learning algorithm [8]. This paper gives a procedure going in the opposite direction.

It is well-known that the online setting is strictly harder than the batch setting, even for the simple one-dimensioanl class of threshold functions on the interval $[0, 1]$. Hence, we consider the online transductive model of Ben-David, Kushilevitz, and Mansour [2]. In this model, an arbitrary but unknown sequence of $n$ examples $(x_1, y_1), \ldots, (x_n, y_n) \in \mathcal{X} \times \{-1, 1\}$ is fixed in advance, for some instance space $\mathcal{X}$. The set of unlabeled examples is then presented to the learner, $\Sigma = \{x_i | 1 \leq i \leq n\}$. The examples are then revealed, in an online manner, to the learner, for $i = 1, 2, \ldots, n$. The learner observes example $x_i$ (along with all previous labeled examples $(x_1, y_1), \ldots, (x_{i-1}, y_{i-1})$ and the unlabeled example set $\Sigma$) and must predict $y_i$. The true label $y_i$ is then revealed to the learner. After this occurs, the learner compares its number of mistakes to the minimum number of mistakes of any of a *target class* $\mathcal{F}$ of functions $f : \mathcal{X} \rightarrow \{-1, 1\}$ (such as linear threshold functions). Note that our results are in this type of *agnostic* model [7], where we allow for arbitrary labels, unlike the *realizable* setting, i.e., noiseless or PAC models, where it is assumed that the labels are consistent with some $f \in \mathcal{F}$.

With this simple *transductive* knowledge of what unlabeled examples are to come, one can use existing expert algorithms to *inefficiently* learn any class of finite VC dimension, similar to the batch setting. How does one use unlabeled examples *efficiently* to guarantee good online performance?

Our efficient algorithm $A_2$ converts a proper[1] batch algorithm to a proper online algorithm (both in the agnostic setting). At any point in time, it has observed some labeled examples. It then "hallucinates" random examples by taking some number of unlabeled examples and labeling them randomly. It appends these examples to those observed so far and predicts according to the batch algorithm that finds the hypothesis of minimum empirical error on the combined data.

The idea of "hallucinating" and optimizing has been used for designing efficient online algorithms [6, 5, 1, 10, 4] in situations where exponential weighting schemes were inefficient. The hallucination analogy was suggested by Blum and Hartline [4]. In the context of transductive learning, it seems to be a natural way to try to use the unlabeled examples in conjunction with a batch learner. Let #mistakes$(f, \sigma_n)$ denote the number of mistakes of a function $f \in \mathcal{F}$ on a particular sequence $\sigma_n \in (\mathcal{X} \times \{-1, 1\})^n$, and #mistakes$(A, \sigma_n)$ denote the same quantity for a transductive online learning algorithm $A$. Our main theorem is the following.

**Theorem 1.** *Let $\mathcal{F}$ be a class of functions $f : \mathcal{X} \to \{-1, 1\}$ of VC dimension $d$. There is an* efficient *randomized transductive online algorithm that, for any $n > 1$ and $\sigma_n \in (\mathcal{X} \times \{-1, 1\})^n$,*

$$\mathbf{E}[\textit{\#mistakes}(A_2, \sigma_n)] \leq \min_{f \in \mathcal{F}} \textit{\#mistakes}(f, \sigma_n) + 2.5 n^{3/4} \sqrt{d \log n}.$$

*The algorithm is computationally efficient in the sense that it runs in time* $\mathrm{poly}(n)$*, given an efficient proper batch learning algorithm.*

One should note that the bound on the error *rate* is the same as that of the best $f \in \mathcal{F}$ plus $O(n^{-1/4} \sqrt{d \log(n)})$, approaching 0 at a rate related to the standard VC bound.

It is well-known that, without regard to computational efficiency, the learnable classes of functions are exactly those with finite VC dimension. Consequently, the classes of functions learnable in the batch and transductive online settings are the same. The classes of functions properly learnable by computationally efficient algorithms in the proper batch and transductive online settings are identical, as well.

In addition to the new algorithm, this is interesting because it helps justify a long line of work suggesting that whatever can be done in a batch setting can also be done online. Our result is surprising in light of earlier work by Blum showing that a slightly different online model is harder than its batch analog for *computational reasons* and not information-theoretic reasons [3].

In Section 2, we define the transductive online model. In Section 3, we analyze the easier case of data that is realizable with respect to some function class, i.e., when there is some function of zero error in the class. In Section 4, we present and analyze the hallucination algorithm. In Section 5, we discuss open problems such as extending the results to improper learning and the efficient realizable case.

## 2 Models and definitions

The *transductive online model* considered by Ben-David, Kushlevitz, and Mansour [2], consists of an instance space $\mathcal{X}$ and label set $\mathcal{Y}$ which we will always take to be binary $\mathcal{Y} = \{-1, 1\}$. An arbitrary $n > 0$ and arbitrary sequence of labeled examples $(x_1, y_1), \ldots, (x_n, y_n)$ is fixed. One can think of these as being chosen by an adversary who knows the (possibly randomized) learning algorithm but not the realization of its random coin flips. For notational convenience, we define $\sigma_i$ to be the subsequence of first $i$

labeled examples,

$$\sigma_i = (x_1, y_1), (x_2, y_2), \ldots, (x_i, y_i),$$

and $\Sigma$ to be the set of all unlabeled examples in $\sigma_n$,

$$\Sigma = \{x_i \mid i \in \{1, 2, \ldots, n\}\}.$$

A transductive online learner $A$ is a function that takes as input $n$ (the number of examples to be predicted), $\Sigma \subseteq \mathcal{X}$ (the set of unlabeled examples, $|\Sigma| \leq n$), $x_i \in \Sigma$ (the example to be tested), and $\sigma_{i-1} \in (\Sigma \times \mathcal{Y})^{i-1}$ (the previous $i - 1$ labeled examples) and outputs a prediction $\in \mathcal{Y}$ of $y_i$, for any $1 \leq i \leq n$. The number of mistakes of $A$ on the sequence $\sigma_n = (x_1, y_1), \ldots, (x_n, y_n)$ is,

$$\#\text{mistakes}(A, \sigma_n) = |\{i \mid A(n, \Sigma, x_i, \sigma_{i-1}) \neq y_i\}|.$$

If $A$ is computed by a randomized algorithm, then we similarly define $\mathbf{E}[\#\text{mistakes}(A, \sigma_n)]$ where the expectation is taken over the random coin flips of $A$. In order to speak of the learnability of a set $\mathcal{F}$ of functions $f : \mathcal{X} \to \mathcal{Y}$, we define

$$\#\text{mistakes}(f, \sigma_n) = |\{i \mid f(x_i) \neq y_i\}|.$$

Formally, paralleling agnostic learning [7],[2] we define an *efficient* transductive online learner $A$ for class $\mathcal{F}$ to be one for which the learning algorithm runs in time $\text{poly}(n)$ and achieves, for any $\epsilon > 0$,

$$\mathbf{E}[\#\text{mistakes}(A, \sigma_n)] \leq \min_{f \in \mathcal{F}} \#\text{mistakes}(f, \sigma_n) + \epsilon n,$$

for $n = \text{poly}(1/\epsilon)$.[3]

## 2.1 Proper learning

Proper batch learning requires one to output a hypothesis $h \in \mathcal{F}$. An efficient proper batch learning algorithm for $\mathcal{F}$ is a batch learning algorithm $B$ that, given any $\epsilon > 0$, with $n = \text{poly}(1/\epsilon)$ many examples from any distribution $\mathcal{D}$, outputs an $h \in \mathcal{F}$ of expected error $\mathbf{E}[\Pr_{\mathcal{D}}[h(x) \neq y]] \leq \min_{f \in \mathcal{F}} \Pr_{\mathcal{D}}[f(x) \neq y] + \epsilon$ and runs in time $\text{poly}(n)$.

**Observation 1.** *Any efficient proper batch learning algorithm $B$ can be converted into an efficient empirical error minimizer $M$ that, for any $n$, given any data set $\sigma_n \in (\mathcal{X} \times \mathcal{Y})^n$, outputs an $f \in \mathcal{F}$ of minimal empirical error on $\sigma_n$.*

*Proof.* Running $B$ only on $\sigma_n$, $B$ is not guaranteed to output a hypothesis of minimum empirical error. Instead, we set an error tolerance of $B$ to $\epsilon = 1/(4n)$, and give it examples drawn uniformly from the distribution $\mathcal{D}$ which is uniform over the data $\sigma_n$ (a type of bootstrap). If $B$ indeed returns a hypothesis $h$ of error less than $1/n$ more than the best $f \in \mathcal{F}$, it must be a hypothesis of minimum empirical error on $\sigma_n$. By Markov's inequality, with probability at most $1/4$, the generalization error is more than $1/n$. By repeating several times and take the best hypothesis, we get a success probability exponentially close to 1. The runtime is polynomial in $n$. $\square$

To define *proper* learning in an online setting, it is helpful to think of the following alternative definition of transductive online learning. In this variation, the learner must output a sequence of hypotheses $h_1, h_2, \ldots, h_n : \mathcal{X} \to \{-1, 1\}$. After the $i$th hypothesis $h_i$ is output, the example $(x_i, y_i)$ is revealed, and it is clear whether the learner made an error. Formally, the (possibly randomized) algorithm $A'$ still takes as input $n$, $\Sigma$, and $\sigma_{i-1}$ (but

no longer $x_i$), and outputs $h_i : \mathcal{X} \rightarrow \{-1, 1\}$ and errs if $h_i(x_i) \neq y_i$. To see that this model is equivalent to the previous definition, note that any algorithm $A'$ that outputs hypotheses $h_i$ can be used to make predictions $h_i(x_i)$ on example $i$ (it errs if $h_i(x_i) \neq y_i$). It is equally true but less obvious than any algorithm $A$ in the previous model can be converted to an algorithm $A'$ in this model. This is because $A'$ can be viewed as outputting $h_i : \mathcal{X} \rightarrow \{-1, 1\}$, where the function $h_i$ is defined by setting $h_i(x)$ equal to be the prediction of algorithm $A$ on the sequence $\sigma_{i-1}$ followed by the example $x$, for each $x \in \mathcal{X}$, i.e., $h_i(x) = A(n, \Sigma, x, \sigma_{i-1})$. (The same coins can be used if $A$ and $A'$ are randomized.) A (possibly randomized) transductive online algorithm in this model is defined to be *proper* for family of functions $\mathcal{F}$ if it always outputs $h_i \in \mathcal{F}$.

## 3   Warmup: the realizable case

In this section, we consider the *realizable* special case in which there is some $f \in \mathcal{F}$ which correctly labels all examples. In particular, this means that we only consider sequences $\sigma_n$ for which there is an $f \in \mathcal{F}$ with #mistakes$(f, \sigma_n) = 0$. This case will be helpful to analyze first as it is easier.

Fix arbitrary $n > 0$ and $\Sigma = \{x_1, x_2, \ldots, x_n\} \subseteq \mathcal{X}$, $|\Sigma| \leq n$. Say there are at most $L$ different ways to label the examples in $\Sigma$ according to functions $f \in \mathcal{F}$, so $1 \leq L \leq 2^{|\Sigma|}$. In the transductive online model, $L$ is determined by $\Sigma$ and $\mathcal{F}$ only. Hence, as long as prediction occurs only on examples $x \in \Sigma$, there are effectively only $L$ different functions in $\mathcal{F}$ that matter, and we can thus pick $L$ such functions that give rise to the $L$ different labelings. On the $i$th example, one could simply take majority vote of $f_j(x_i)$ over consistent labelings $f_j$ (the so-called *halving algorithm*), and this would easily ensure at most $\log_2(L)$ mistakes, because each mistake eliminates at least half of the consistent labelings. One can also use the following proper learning algorithm.

**Proper transductive online learning algorithm in the realizable case:**

- Preprocessing: Given the set of unlabeled examples $\Sigma$, take $L$ functions $f_1, f_2, \ldots, f_L \in \mathcal{F}$ that give rise to the $L$ different labelings of $x \in \Sigma$.[4]
- $i$th prediction: Output a uniformly random function $f$ from the $f_j$ consistent with $\sigma_{i-1}$.

The above algorithm, while possibly very inefficient, is easy to analyze.

**Theorem 2.** *Fix a class of binary functions $\mathcal{F}$ of VC dimension $d$. The above randomized proper learning algorithm makes an expected $d \log(n)$ mistakes on any sequence of examples of length $n \geq 2$, provided that there is some mistake-free $f \in \mathcal{F}$.*

*Proof.* Let $V_i$ be the number of labelings $f_j$ consistent with the first $i$ examples, so that $L = V_0 \geq V_1 \geq \cdots \geq V_n \geq 1$ and $L \leq n^d$, by Sauer's lemma [11] for $n \geq 2$, where $d$ is the VC dimension of $\mathcal{F}$. Observe that the number of consistent labelings that make a mistake on the $i$th example are exactly $V_{i-1} - V_i$. Hence, the total expected number of mistakes is,

$$\sum_{i=1}^{n} \frac{V_{i-1} - V_i}{V_{i-1}} \leq \sum_{i=1}^{n} \left( \frac{1}{V_{i-1}} + \frac{1}{V_{i-1} - 1} + \cdots \frac{1}{V_i + 1} \right) \leq \sum_{i=2}^{V_n} \frac{1}{i} \leq \log(L). \qquad \square$$

Hence the above algorithm achieves an error rate of $O(d \log(n)/n)$, which quickly approaches zero for large $n$. Note that, this closely matches what one achieves in the batch setting. Like the batch setting, no better bounds can be given up to a constant factor.

## 4 General setting

We now consider the more difficult unrealizable setting where we have an unconstrained sequence of examples (though we still work in a transductive setting). We begin by presenting an known (inefficnet) extension to the halving algorithm of the previous section, that works in the agnostic (unrealizable) setting that is similar to the previous algorithm.

**Inefficient proper transductive online learning algorithm $A_1$:**

- Preprocessing: Given the set of unlabeled examples $\Sigma$, take $L$ functions $f_1, f_2, \ldots, f_L$ that give rise to the $L$ different labelings of $x \in \Sigma$. Assign an initial *weight* $w_1 = w_2 = \ldots = w_L = 1$ to each function.
- Output $f_j$, where $1 \leq j \leq L$ is chosen with probability $\frac{w_j}{w_1 + \ldots + w_L}$.
- Update: for each $j$ for which $f_j(x_i) \neq y_i$, reduce $w_j$,
$$w_j := w_j \left( 1 - \sqrt{\frac{\log L}{n}} \right).$$

Using an analysis very similar to that of Weighted Majority [9], one can show that, for any $n > 1$ and sequence of examples $\sigma_n \in (\mathcal{X} \times \{-1, 1\})^n$,

$$\mathbf{E}[\#\text{mistakes}(A_1, \sigma_n)] = \min_{f \in \mathcal{F}} \#\text{mistakes}(f, \sigma_n) + 2\sqrt{dn \log n},$$

where $d$ is the VC dimension of $\mathcal{F}$. Note the similarity to the standard VC bound.

### 4.1 Efficient algorithm

We can only hope to get an efficient proper online algorithm when there is an efficient proper batch algorithm. As mentioned in section 2.1, this means that there is a batch algorithm $M$ that, given any data set, efficiently finds a hypothesis $h \in \mathcal{F}$ of minimum empirical error. (In fact, most proper learning algorithms work this way to begin with.) Using this, our efficient algorithm is as follows.

**Efficient transductive online learning algorithm $A_2$:**

- Preprocessing: Given the set of unlabeled examples $\Sigma$, create a hallucinated data set $\tau$ as follows.
  1. For each example $x \in \Sigma$, choose integer $r_x$ uniformly at random such that $-\sqrt[4]{n} \leq r_x \leq \sqrt[4]{n}$.
  2. Add $|r_x|$ copies of the example $x$ labeled by the sign of $r_x$, $(x, \text{sgn}(r_x))$, to $\tau$.
- To predict on $x_i$: output hypothesis $M(\tau\sigma_{i-1}) \in \mathcal{F}$, where $\tau\sigma_{i-1}$ is the concatenation of the hallucinated examples and the observed labeled examples so far.

The current algorithm predicts $f(x_i)$ based on $f = M(\tau\sigma_{i-1})$. We first begin by analyzing the hypothetical algorithm that used the function chosen on the next iteration, i.e. predict $f(x_i)$ based on $f = M(\tau\sigma_i)$. (Of course, this is impossible to implement because we do not know $\sigma_i$ when predicting $f(x_i)$.)

**Lemma 1.** *Fix any $\tau \in (\mathcal{X} \times \mathcal{Y})^*$ and $\sigma_n \in (\mathcal{X} \times \mathcal{Y})^n$. Let $A_2'$ be the algorithm that, for each $i$, predicts $f(x_i)$ based on $f \in \mathcal{F}$ which is any empirical minimizer on the concatenated data $\tau\sigma_i$, i.e., $f = M(\tau\sigma_i)$. Then the total number of mistakes of $A_2'$ is,*

$$\#mistakes(A_2', \sigma_n) \leq \min_{f \in \mathcal{F}} \#mistakes(f, \tau\sigma_n) - \min_{f \in \mathcal{F}} \#mistakes(f, \tau).$$

It is instructive to first consider the case where $\tau$ is empty, i.e., there are no hallucinated examples. Then, our algorithm that predicts according to $M(\sigma_{i-1})$ could be called "follow the leader," as in [6]. The above lemma means that if one could use the hypothetical "be the leader" algorithm then one would make no more mistakes than the best $f \in \mathcal{F}$. The proof of this case is simple. Imagine starting with the offline algorithm that uses $M(\sigma_n)$ on each example $x_1, \ldots, x_n$. Now, on the first $n-1$ examples, replace the use of $M(\sigma_n)$ by $M(\sigma_{n-1})$. Since $M(\sigma_{n-1})$ is an error-minimizer on $\sigma_{n-1}$, this can only reduce the number of mistakes. Next replace $M(\sigma_{n-1})$ by $M(\sigma_{n-2})$ on the first $n-2$ examples, and so on. Eventually, we reach the hypothetical algorithm above, and we have only decreased our number of mistakes. The proof of the above lemma follows along these lines.

*Proof of Lemma 1.* Fix empirical minimizers $g_i$ on $\tau\sigma_i$ for $i = 0, 1, \ldots, n$, i.e., $g_i = M(\tau\sigma_i)$. For $i \geq 1$, let $m_i$ be 1 if $g_i(x_j) \neq y_j$ and 0 otherwise. We argue by induction on $t$ that,

$$\#mistakes(g_0, \tau) + \sum_{i=1}^{t} m_i \leq \#mistakes \text{ of } g_t \text{ on } \tau\sigma_t. \tag{1}$$

For $t = 0$, the two are trivially equal. Assuming it holds for $t$, we have,

$$
\begin{aligned}
\#mistakes(g_0, \tau) + \sum_{i=1}^{t+1} m_i &\leq \#mistakes(g_t, \tau\sigma_t) + m_{t+1} \\
&\leq \#mistakes(g_{t+1}, \tau\sigma_t) + m_{t+1} \\
&= \#mistakes(g_{t+1}, \tau\sigma_{t+1}).
\end{aligned}
$$

The first inequality above holds by induction hypothesis, and the second follows from the fact that $g_t$ is an empirical minimizer of $\tau\sigma_t$. The equality establishes (1) for $t+1$ and thus completes the induction. The total mistakes of the hypothetical algorithm proposed in the lemma is $\sum_{i=1}^{n} m_i$, which gives the lemma by rearranging (1) for $t = n$. $\qquad\square$

**Lemma 2.** *For any $\sigma_n$,*

$$\mathbf{E}_\tau[\min_{f \in \mathcal{F}} \#mistakes(f, \tau\sigma_n)] \leq \mathbf{E}_\tau[|\tau|/2] + \min_{f \in \mathcal{F}} \#mistakes(f, \sigma_n).$$

*For any $\mathcal{F}$ of VC dimension $d$,*

$$\mathbf{E}_\tau[\min_{f \in \mathcal{F}} \#mistakes(f, \tau)] \geq \mathbf{E}_\tau[|\tau|/2] - 1.5 n^{3/4}\sqrt{d \log n}.$$

*Proof.* For the first part of the lemma, let $g = M(\sigma_n)$ be an empirical minimizer on $\sigma_n$. Then,

$$\mathbf{E}_\tau[\min_{f \in \mathcal{F}} \#mistakes(f, \tau\sigma_n)] \leq E_\tau[\#mistakes(g, \tau\sigma_n)] = \mathbf{E}_\tau[|\tau|/2] + \#mistakes(g, \sigma_n).$$

The last inequality holds because, since each example in $\tau$ is equally likely to have a $\pm$ label, the expected number of mistakes of any fixed $g \in \mathcal{F}$ on $\tau$ is $\mathbf{E}[|\tau|/2]$.

Fix any $f \in \mathcal{F}$. For the second part of the lemma, observe that we can write the number of mistakes of $f$ on $\tau$ as,

$$\#mistakes(f, \tau) = \frac{|\tau| - \sum_{i=1}^{n} f(x_i) r_i}{2}.$$

Hence it suffices to show that, $\max_{f \in \mathcal{F}} \sum_{i=1}^{n} f(x_i)r_i \leq 3n^{3/4}\sqrt{\log(L)}$.

Now $\mathbf{E}_{r_i}[f(x_i)r_i] = 0$ and $|f(x_i)r_i| \leq n^{1/4}$. Next, Chernoff bounds (on the scaled random variables $f(x_i)r_i n^{-1/4}$) imply that, for any $\alpha \leq 1$, with probability at most $e^{-n\alpha^2/2}$, $\sum_{i=1}^{n} f(x_i)r_i n^{-1/4} \geq n\alpha$. Put another way, for any $\beta < n$, with probability at most $e^{-n^{-3/2}\beta^2/2}$, $\sum f(x_i)r_i n^{-1/4} \geq \beta$. As observed before, we can reduce the problem to the $L$ different labelings. In other words, we can assume that there are only $L$ different functions. By the union bound, the probability that $\sum f(x_i)r_i \geq \beta$ for any $f \in \mathcal{F}$ is at most $Le^{-n^{-3/2}\beta^2/2}$. Now the expectation of a non-negative random variable $X$ is $\mathbf{E}[X] = \int_0^{\infty} \Pr[X \geq x]dx$. Let $X = \max_{f \in \mathcal{F}} \sum_{i=1}^{n} f(x_i)r_i$. In our case,

$$\mathbf{E}[X] \leq \sqrt{2\log(L)}n^{3/4} + \int_{\sqrt{2\log(L)}n^{3/4}}^{\infty} Le^{-n^{-3/4}x^2/2}dx$$

By Mathematica, the above is at most $\sqrt{2\log(L)}n^{3/4} + 1.254n^{3/4} \leq 3\sqrt{\log(L)}n^{3/4}$. Finally, we use the fact that $L \leq n^d$ by Sauer's lemma. □

Unfortunately, we cannot use the algorithm $A_2'$. However, due to the randomness we have added, we can argue that algorithm $A_2$ is quite close:

**Lemma 3.** *For any $\sigma_n$, for any $i$, with probability at least $1 - n^{-1/4}$ over $\tau$, $M(\tau\sigma_{i-1})$ is an empirical minimizer of $\tau\sigma_i$.*

*Proof.* Define, $\mathcal{F}_+ = \{f \in \mathcal{F} \mid f(x_i) = 1\}$ and $\mathcal{F}_- = \{f \in \mathcal{F} \mid f(x_i) = -1\}$. WLOG, we may assume that $\mathcal{F}_+$ and $\mathcal{F}_-$ are both nonempty. For if not, i.e., if all $f \in \mathcal{F}$ predict the same sign $f(x_i)$, then the sets of empirical minimizers of $\tau\sigma_{i-1}$ and $\tau\sigma_i$ are equal and the lemma holds trivially. For any sequence $\pi \in (\mathcal{X} \times \mathcal{Y})^*$, define,

$$s_+(\pi) = \min_{f \in \mathcal{F}_+} \#\text{mistakes}(f, \pi) \text{ and } s_-(\pi) = \min_{f \in \mathcal{F}_-} \#\text{mistakes}(f, \pi).$$

Next observe that, if $s_+(\pi) < s_-(\pi)$ then $M(\pi) \in \mathcal{F}_+$. Similarly if $s_-(\pi) < s_+(\pi)$ then $M(\pi) \in \mathcal{F}_-$. If they are equal then $f(x_i)$ can be an empirical minimizer in either. WLOG let us say that the $i$th example is $(x_i, 1)$, i.e., it is labeled positively. This implies that $s_+(\tau\sigma_{i-1}) = s_+(\tau\sigma_i)$ and $s_-(\tau\sigma_{i-1}) = s_-(\tau\sigma_i) + 1$. It is now clear that if $M(\tau\sigma_{i-1})$ is not also an empirical minimizer of $\tau\sigma_i$ then $s_+(\tau\sigma_{i-1}) = s_-(\tau\sigma_{i-1})$.

Now the quantity $\Delta = s_+(\tau\sigma_{i-1}) - s_-(\tau\sigma_{i-1})$ is directly related to $r_{x_i}$, the signed random number of times that example $x_i$ is hallucinated. If we fix $\sigma_n$ and the random choices $r_x$ for each $x \in \Sigma \setminus \{x_i\}$, as we increase or decrease $r_i$ by 1, $\Delta$ correspondingly increases or decreases by 1. Since $r_i$ was chosen from a range of size $2\lfloor n^{1/4} \rfloor + 1 \geq n^{1/4}$, $\Delta = 0$ with probability at most $n^{-1/4}$. □

We are now ready to prove the main theorem.

*Proof of Theorem 1.* Combining Lemmas 1 and 2, if on each period $i$, we used any minimizer of empirical error on the data $\tau\sigma_i$, we would have a total number of mistakes of at most $\min_{f \in \mathcal{F}} \#\text{mistakes}(f, \sigma_n) + 1.5n^{3/4}\sqrt{d\log n}$. Suppose $A_2$ does end up using such a minimizer on all but $p$ periods. Then, its total number of mistakes can only be $p$ larger than this bound. By Lemma 3, the expected number $p$ of periods $i$ in which an empirical minimizer of $\tau\sigma_i$ is not used is $\leq n^{3/4}$. Hence, the expected total number of mistakes of $A_2$ is at most,

$$\mathbf{E}_\tau[\#\text{mistakes}(A_2, \sigma_n)] \leq \min_{f \in \mathcal{F}} \#\text{mistakes}(f, \sigma_n) + 1.5n^{3/4}\sqrt{d\log n} + n^{3/4}.$$

The above implies the theorem. □

**Remark 1**. The above algorithm is still costly in the sense that we must re-run the batch error minimizer for each prediction we would like to make. Using an idea quite similar to the "follow the lazy leader" algorithm in [6], we can achieve the same expected error while only needing to call $M$ with probability $n^{-1/4}$ on each example.

**Remark 2**. The above analysis resembles previous analysis of hallucination algorithms. However, unlike previous analyses, there is no exponential distribution in the hallucination here yet the bounds still depend only logarithmically on the number of labelings.

## 5   Conclusions and open problems

We have given an algorithm for learning in the transductive online setting and established several results between efficient proper batch and transductive online learnability. In the realizable case, however, we have not given a computationally efficient algorithm. Hence, it is an open question as to whether *efficient* learnability in the batch and transductive on-line settings are the same in the realizable case. In addition, our computationally efficient algorithm requires polynomially more examples than its inefficient counterpart. It would be nice to have the best of both worlds, namely a computationally efficient algorithm that achieves a number of mistakes that is at most $O(\sqrt{dn\log n})$. Additionally, it would be nice to remove the restriction to proper algorithms.

**Acknowledgements.** We would like to thank Maria-Florina Balcan, Dean Foster, John Langford, and David McAllester for helpful discussions.

## Footnotes

[1]A *proper* learning algorithm is one that always outputs a hypothesis $h \in \mathcal{F}$.

[2]It is more common in online learning to bound the total number of mistakes of an online algorithm on an arbitrary sequence. We bound its error rate, as is usual for batch learning.

[3]The results in this paper could be replaced by high-probability $1 - \delta$ bounds at a cost of $\log 1/\delta$.

[4]More formally, take $L$ functions with the following properties: for each pair $1 \leq j, k \leq L$ with $j \neq k$, there exists $x \in \Sigma$ such that $f_j(x) \neq f_k(x)$, and for every $f \in \mathcal{F}$, there exists a $1 \leq j \leq L$ with $f(x) = f_j(x)$ for all $x \in \Sigma$.

## References

[1]  B. Awerbuch and R. Kleinberg. Adaptive routing with end-to-end feedback: Distributed learning and geometric approaches. In *Proc. of the 36th ACM Symposium on Theory of Computing*, 2004.

[2]  S. Ben-David, E. Kushilevitz, and Y. Mansour. Online learning versus offline learning. *Machine Learning* 29:45-63, 1997.

[3]  A. Blum. Separating Distribution-Free and Mistake-Bound Learning Models over the Boolean Domain. *SIAM Journal on Computing* 23(5): 990-1000, 1994.

[4]  A. Blum, J. Hartline. Near-Optimal Online Auctions. In *Proceedings of the Proceedings of the Sixteenth Annual ACM-SIAM Symposium on Discrete Algorithms* (SODA), 2005.

[5]  J. Hannan. Approximation to Bayes Risk in Repeated Plays. In M. Dresher, A. Tucker, and P. Wolfe editors, *Contributions to the Theory of Games, Volume 3*, p. 97-139, Princeton University Press, 1957.

[6]  A. Kalai and S. Vempala. Efficient algorithms for the online decision problem. In *Proceedings of the 16th Conference on Computational Learning Theory,* 2003.

[7]  M. Kearns, R. Schapire, and L. Sellie. Toward Efficient Agnostic Learning. *Machine Learning*, 17(2/3):115–141, 1994.

[8]  N. Littlestone. From On-Line to Batch Learning. In *Proceedings of the 2nd Workshop on Computational Learning Theory*, p. 269-284, 1989.

[9]  N. Littlestone and M. Warmuth. The Weighted Majority Algorithm. *Information and Computation,* 108:212-261, 1994.

[10]  H. Brendan McMahan and Avrim Blum. Online Geometric Optimization in the Bandit Setting Against an Adaptive Adversary. In *Proceedings of the 17th Annual Conference on Learning Theory*, COLT 2004.

[11]  N. Sauer. On the Densities of Families of Sets. *Journal of Combinatorial Theory, Series A*, 13, p 145-147, 1972.

[12]  V. N. Vapnik. *Estimation of Dependencies Based on Empirical Data,* New York: Springer Verlag, 1982.

[13]  V. N. Vapnik. *Statistical Learning Theory*, New York: Wiley Interscience, 1998.
